# Tree-structured approximations by expectation propagation

**Thomas Minka**
Department of Statistics
Carnegie Mellon University
Pittsburgh, PA 15213 USA
minka@stat.cmu.edu

**Yuan Qi**
Media Laboratory
Massachusetts Institute of Technology
Cambridge, MA 02139 USA
yuanqi@media.mit.edu

## Abstract

Approximation structure plays an important role in inference on loopy graphs. As a tractable structure, tree approximations have been utilized in the variational method of Ghahramani & Jordan (1997) and the sequential projection method of Frey et al. (2000). However, belief propagation represents each factor of the graph with a product of single-node messages. In this paper, belief propagation is extended to represent factors with tree approximations, by way of the expectation propagation framework. That is, each factor sends a "message" to all pairs of nodes in a tree structure. The result is more accurate inferences and more frequent convergence than ordinary belief propagation, at a lower cost than variational trees or double-loop algorithms.

## 1 Introduction

An important problem in approximate inference is improving the performance of belief propagation on loopy graphs. Empirical studies have shown that belief propagation (BP) tends not to converge on graphs with strong positive and negative correlations (Welling & Teh, 2001). One approach is to force the convergence of BP, by appealing to a free-energy interpretation (Welling & Teh, 2001; Teh & Welling, 2001; Yuille, In press 2002). Unfortunately, this doesn't really solve the problem because it dramatically increases the computational cost and doesn't necessarily lead to good results on these graphs (Welling & Teh, 2001).

The expectation propagation (EP) framework (Minka, 2001a) gives another interpretation of BP, as an algorithm which approximates multi-variable factors by single-variable factors $(f(x_1, x_2) \rightarrow \tilde{f}_1(x_1)\tilde{f}_2(x_2))$. This explanation suggests that it is BP's target approximation which is to blame, not the particular iterative scheme it uses. Factors which encode strong correlations should not be well approximated in this way. The connection between failure to converge and poor approximation holds true for EP algorithms in general, as shown by Minka (2001a) and Heskes & Zoeter (2002).

Yedidia et al. (2000) describe an extension of BP involving the Kikuchi free-energy. The resulting algorithm resembles BP on a graph of node clusters, where again multi-variable factors are decomposed into independent parts $(f(x_1, x_2, x_3) \rightarrow \tilde{f}_1(x_1)\tilde{f}_{23}(x_2, x_3))$.

In this paper, the target approximation of BP is enriched by exploiting the connection to expectation propagation. Instead of approximating each factor by disconnected nodes or clusters, it is approximated by a tree distribution. The algorithm is a strict generalization of belief propagation, because if the tree has no edges, then the results are identical to (loopy) belief propagation.

This approach is inspired by previous work employing trees. For example, Ghahramani & Jordan (1997) showed that tree structured approximations could improve the accuracy of variational bounds. Such bounds are tuned to minimize the 'exclusive' KL-divergence $KL(q||p)$, where $q$ is the approximation. Frey et al. (2000) criticized this error measure and described an alternative method for minimizing the 'inclusive' divergence $KL(p||q)$. Their method, which sequentially projects graph potentials onto a tree structure, is closely related to expectation propagation and the method in this paper. However, their method is not iterative and therefore sensitive to the order in which the potentials are sequenced.

There are also two tangentially related papers by Wainwright et al. (2001); Wainwright et al. (2002). In the first paper, a "message-free" version of BP was derived, which used multiple tree structures to propagate evidence. The results it gives are nevertheless the same as BP. In the second paper, tree structures were used to obtain an upper bound on the normalizing constant of a Markov network. The trees produced by that method do not necessarily approximate the original distribution well.

The following section describes the EP algorithm for updating the potentials of a tree approximation with known structure. Section 3 then describes the method we use to choose the tree structure. Section 4 gives numerical results on various graphs, comparing the new algorithm to BP, Kikuchi, and variational methods.

## 2   Updating the tree potentials

This section describes an expectation-propagation algorithm to approximate a given distribution (of arbitrary structure) by a tree with known structure. It elaborates section 4.2.2 of Minka (2001b), with special attention to efficiency. Denote the original distribution by $p(\mathbf{x})$, written as a product of factors:

$$p(\mathbf{x}) = \prod_i f_i(\mathbf{x}) \tag{1}$$

For example, if $p(\mathbf{x})$ is a Bayesian network or Markov network, the factors are conditional probability distributions or potentials which each depend on a small subset of the variables in $\mathbf{x}$. In this paper, the variables are assumed to be discrete, so that the factors $f_i(\mathbf{x})$ are simply multidimensional tables.

### 2.1   Junction tree representation

The target approximation $q(\mathbf{x})$ will have pairwise factors along a tree $\mathcal{T}$:

$$q(\mathbf{x}) = \frac{\prod_{(j,k)\in\mathcal{T}} q(x_j, x_k)}{\prod_{s\in\mathcal{S}} q(x_s)} \tag{2}$$

In this notation, $q(x_s)$ is the marginal distribution for variable $x_s$ and $q(x_j, x_k)$ is the marginal distribution for the two variables $x_j$ and $x_k$. These are going to be stored as multidimensional tables. The division is necessary to cancel over-counting in the numerator. A useful way to organize these divisions is to construct a *junction tree* connecting the cliques $(j, k) \in \mathcal{T}$ (Jensen et al., 1990). This tree has a different structure than $\mathcal{T}$—the nodes in the junction tree represent cliques in $\mathcal{T}$, and the edges in the junction tree represent variables which are shared between cliques. These *separator* variables $\mathcal{S}$ in the junction tree are

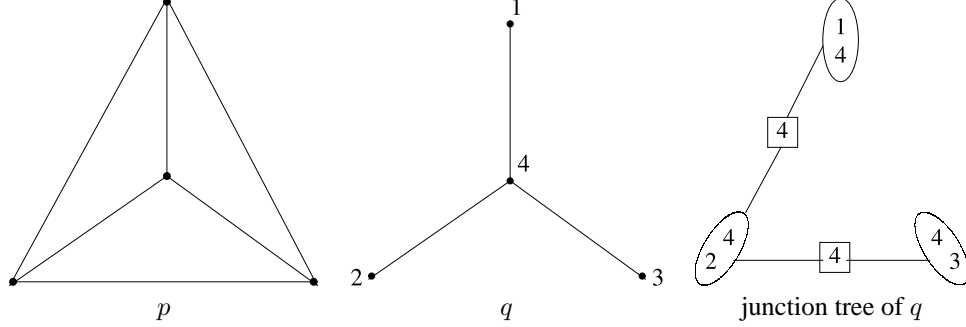

Figure 1: Approximating a complete graph $p$ by a tree $q$. The junction tree of $q$ is used to organize computations.

exactly the variables that go in the denominator of (2). Note that the same variable could be a separator more than once, so technically $\mathcal{S}$ is a multiset.

Figure 1 shows an example of how this all works. We want to approximate the distribution $p(\mathbf{x})$, which has a complete graph, by $q(\mathbf{x})$, whose graph is a spanning tree. The marginal representation of $q$ can be directly read off of the junction tree:

$$q(\mathbf{x}) = \frac{q(x_1, x_4)q(x_2, x_4)q(x_3, x_4)}{q(x_4)q(x_4)} \tag{3}$$

## 2.2 EP updates

The algorithm iteratively tunes $q(\mathbf{x})$ so that it matches $p(\mathbf{x})$ as closely as possible, in the sense of 'inclusive' KL-divergence. Specifically, $q$ tries to preserve the marginals and pairwise marginals of $p$:

$$q(x_j) \approx p(x_j) \tag{4}$$
$$q(x_j, x_k) \approx p(x_j, x_k) \qquad (j, k) \in \mathcal{T} \tag{5}$$

Expectation propagation is a general framework for approximating distributions of the form (1) by approximating the factors one by one. The final approximation $q$ is then the product of the approximate factors. The functional form of the approximate factors is determined by considering the ratio of two different $q$'s. In our case, this leads to approximations of the form

$$f_i(\mathbf{x}) \approx \tilde{f}_i(\mathbf{x}) = \frac{\prod_{(j,k)\in\mathcal{T}} \tilde{f}_i(x_j, x_k)}{\prod_{s\in\mathcal{S}} \tilde{f}_i(x_s)} \tag{6}$$

A product of such factors gives a distribution of the desired form (2). Note that $\tilde{f}_i(x_j, x_k)$ is not a proper marginal distribution, but just a non-negative function of two variables.

The algorithm starts by initializing the clique and separator potentials on the junction tree to 1. If a factor in $p$ only depends on one variable, or variables which are adjacent in $\mathcal{T}$, then its approximation is trivial. It can be multiplied into the corresponding clique potential right away and removed from further consideration. The remaining factors in $p$, the *off-tree* factors, have their approximations $\tilde{f}_i$ initialized to 1.

To illustrate, consider the graph of figure 1. Suppose all the potentials in $p$ are pairwise, one for each edge. The edges $\{(1,4), (2,4), (3,4)\}$ are absorbed directly into $q$. The off-tree edges are $\{(1,2), (1,3), (2,3)\}$.

The algorithm then iteratively passes through the off-tree factors in $p$, performing the following three steps until all $\tilde{f}_i$ converge:

(a) *Deletion.* Remove $\tilde{f}_i$ from $q$ to get an 'old' approximation $q^{\backslash i}$:

$$q^{\backslash i}(x_j, x_k) \quad = \quad \frac{q(x_j, x_k)}{\tilde{f}_i(x_j, x_k)} \qquad (j, k) \in \mathcal{T} \qquad\qquad (7)$$

$$q^{\backslash i}(x_s) \quad = \quad \frac{q(x_s)}{\tilde{f}_i(x_s)} \qquad s \in \mathcal{S} \qquad\qquad (8)$$

(b) *Incorporate evidence.* Form the product $f_i(\mathbf{x})q^{\backslash i}(\mathbf{x})$, by considering $f(\mathbf{x})$ as 'evidence' for the junction tree. Propagate the evidence to obtain new clique marginals $q(x_j, x_k)$ and separators $q(x_s)$ (details below).

(c) *Update.* Re-estimate $\tilde{f}_i$ by division:

$$\tilde{f}_i(x_j, x_k) \quad = \quad \frac{q(x_j, x_k)}{q^{\backslash i}(x_j, x_k)} \qquad (j, k) \in \mathcal{T} \qquad\qquad (9)$$

$$\tilde{f}_i(x_s) \quad = \quad \frac{q(x_s)}{q^{\backslash i}(x_s)} \qquad s \in \mathcal{S} \qquad\qquad (10)$$

## 2.3   Incorporating evidence by cutset conditioning

The purpose of the "incorporate evidence" step is to find a distribution $q$ minimizing $KL(f_i(\mathbf{x})q^{\backslash i} \,\|\, q)$. This is equivalent to matching the marginal distributions corresponding to each clique in $q$. By definition, $f_i$ depends on a set of variables which are not adjacent in $\mathcal{T}$, so the graph structure corresponding to $f_i(\mathbf{x})q^{\backslash i}(\mathbf{x})$ is not a tree, but has one or more loops. One approach is to apply a generic exact inference algorithm to $f_i(\mathbf{x})q^{\backslash i}(\mathbf{x})$ to obtain the desired marginals, e.g. construct a new junction tree in which $f_i(\mathbf{x})$ is a clique and propagate evidence in this tree. But this does not exploit the fact that we already have a junction tree for $q^{\backslash i}$ on which we can perform efficient inference.

Instead we use a more efficient approach—Pearl's cutset conditioning algorithm—to incorporate the evidence. Suppose $f_i(\mathbf{x})$ depends on a set of variables $\mathcal{V}$. The *domain* of $f_i(\mathbf{x})$ is the set of all possible assignments to $\mathcal{V}$. Find the clique $(j, k) \in \mathcal{T}$ which has the largest overlap with this domain—call this the *root clique*. Then enumerate the rest of the domain $\mathcal{V}\backslash(x_j, x_k)$. For each possible assignment to these variables, enter it as evidence in $q$'s junction tree and propagate to get marginals and an overall scale factor (which is the probability of that assignment). When the variables $\mathcal{V}\backslash(x_j, x_k)$ are fixed, entering evidence simply reduces to zeroing out conflicting entries in the junction tree, and multiplying the root clique $(j, k)$ by $f_i(\mathbf{x})$. After propagating evidence multiple times, average the results together according to their scale factors, to get the final marginals and separators of $q$.

Continuing the example of figure 1, suppose we want to process edge $(1, 2)$, whose factor is $f_1(x_1, x_2)$. When added to $q$, this creates a loop. We cut the loop by conditioning on the variable with smallest arity. Suppose $x_1$ is binary, so we condition on it. The other clique, $(2, 4)$, becomes the root. In one case, the evidence is $(x_1 = 0, f_1(0, x_2))$ and in the other it is $(x_1 = 1, f_1(1, x_2))$. Propagating evidence for both cases and averaging the results gives the new junction tree potentials.

Because it is an expectation-propagation algorithm, we know that a fixed point always exists, but we may not always find one. In these cases, the algorithm could be stabilized by a stepsize or double-loop iteration. But overall the method is very stable, and in this paper no convergence control is used.

### 2.4 Within-loop propagation

A further optimization is also used, by noting that evidence does not need to be propagated to the whole junction tree. In particular, it only needs to be propagated within the subtree that connects the nodes in $\mathcal{V}$. Evidence propagated to the rest of the tree will be exactly canceled by the separators, so even though the potentials may change, the ratios in (2) will not. For example, when we process edge $(1, 2)$ in figure 1, there is no need to propagate evidence to clique $(3, 4)$, because when $q(x_3, x_4)$ is divided by the separator $q(x_4)$, we have $q(x_3|x_4)$ which is the same before and after the evidence.

Thus evidence is propagated as follows: first collect evidence from $\mathcal{V}$ to the root, then distribute evidence from the root back to $\mathcal{V}$, bypassing the rest of the tree (these operations are defined formally by Jensen et al. (1990)). In the example, this means we collect evidence from clique $(1, 4)$ to the root $(2, 4)$, then distribute back from $(2, 4)$ to $(1, 4)$, ignoring $(3, 4)$. This simplification also means that we don't need to store $\tilde{f}_i$ for the cliques that are never updated by factor $i$. When moving to the next factor, once we've designated the root for that factor, we collect evidence from the previous root. In this way, the results are the same as if we always propagated evidence to the whole junction tree.

## 3 Choosing the tree structure

This section describes a simple method to choose the tree structure. It leaves open the problem of finding the 'optimal' approximation structure; instead, it presents a simple rule which works reasonably well in practice.

Intuitively, we want edges between the variables which are the most correlated. The approach is based on Chow & Liu (1968): estimate the mutual information between adjacent nodes in $p$'s graph, call this the 'weight' of the edge between them, and then find the spanning tree with maximal total weight. The mutual information between nodes requires an estimate of their joint distribution. In our implementation, this is obtained from the product of factors involving only these two nodes, i.e. the single-node potentials times the edge between them. While crude, it does capture the amount of correlation provided by the edge, and thus whether we should have it in the approximation.

## 4 Numerical results

### 4.1 The four-node network

This section illustrates the algorithm on a concrete problem, comparing it to other methods for approximate inference. The network and approximation will be the ones pictured in figure 1, with all nodes binary. The potentials were chosen randomly and can be obtained from the authors' website.

Five approximate inference methods were compared. The proposed method (TreeEP) used the tree structure specified in figure 1. Mean-field (MF) fit a variational bound with independent variables, and TreeVB fit a tree-structured variational bound, with the same structure as TreeEP. TreeVB was implemented using the general method described by Wiegerinck (2000), with the same junction tree optimizations as in TreeEP.

Generalized belief propagation (GBP) was implemented using the parent-child algorithm of Yedidia et al. (2002) (with special attention to the damping described in section 8). We also used GBP to perform ordinary loopy belief propagation (BP). Our implementation tries to be efficient in terms of FLOPS, but we do not know if it is the fastest possible. GBP and BP were first run using stepsize $0.5$, and if didn't converge, halved it and started over. The time for these 'trial runs' was not counted.

| Method | FLOPS | $E[x_1]$ | $E[x_2]$ | $E[x_3]$ | $E[x_4]$ | Error |
|--------|-------|----------|----------|----------|----------|-------|
| Exact | 200 | 0.474 | 0.468 | 0.482 | 0.536 | 0 |
| TreeEP | 800 | 0.467 | 0.459 | 0.477 | 0.535 | 0.008 |
| GBP | 2200 | 0.467 | 0.459 | 0.477 | 0.535 | 0.008 |
| TreeVB | 11700 | 0.460 | 0.460 | 0.476 | 0.540 | 0.014 |
| BP | 500 | 0.499 | 0.499 | 0.5 | 0.501 | 0.035 |
| MF | 11500 | 0.000 | 0.000 | 0.094 | 0.946 | 0.474 |

Table 1: Node means estimated by various methods (TreeEP = the proposed method, BP = loopy belief propagation, GBP = generalized belief propagation on triangles, MF = mean-field, TreeVB = variational tree). FLOPS are rounded to the nearest hundred.

The algorithms were all implemented in Matlab using Kevin Murphy's BNT toolbox (Murphy, 2001). Computational cost was measured by the number of floating-point operations (FLOPS). Because the algorithms are iterative and can be stopped at any time to get a result, we used a "5% rule" to determine FLOPS. The algorithm was run for a large number of iterations, and the error at each iteration was computed. At each iteration, we then get an error *bound*, which is the maximum error from that iteration onwards. The first iteration whose error bound is within 5% of the final error is chosen for the official FLOP count. (The official error is still the final error.)

The results are shown in table 1. TreeEP is more accurate than BP, with less cost than TreeVB and GBP. GBP was run with clusters $\{(1,2,4),(1,3,4),(2,3,4)\}$. This gives the same result as TreeEP, because these clusters are exactly the off-tree loops.

## 4.2 Complete graphs

The next experiment tests the algorithms on complete graphs of varying size. The graphs have random single-node and pairwise potentials, of the form $f_i(x_j) = [\exp(\theta_j) \exp(-\theta_j)]$ and $f_i(x_j, x_k) = \begin{bmatrix} \exp(w_{jk}) & \exp(-w_{jk}) \\ \exp(-w_{jk}) & \exp(w_{jk}) \end{bmatrix}$. The "external fields" $\theta_j$ were drawn independently from a Gaussian with mean 0 and standard deviation 1. The "couplings" $w_{jk}$ were drawn independently from a Gaussian with mean 0 and standard deviation $3/\sqrt{n-1}$, where $n$ is the number of nodes. Each node has $n-1$ neighbors, so this tries to keep the overall coupling level constant.

Figure 2(a) shows the approximation error as $n$ increases. For each $n$, 10 different potentials were drawn, giving 110 networks in all. For each one, the maximum absolute difference between the estimated means and exact means was computed. These errors are averaged over potentials and shown separately for each graph size. TreeEP and TreeVB always used the same structure, picked according to section 3. TreeEP outperforms BP consistently, but TreeVB does not.

For this type of graph, we found that GBP works well with clusters in a 'star' pattern, i.e. the clusters are $\{(1,2,3),(1,3,4),(1,4,5),...,(1,n,2)\}$. Node '1' is the center of the star, and was chosen to be the node with highest average coupling to its neighbors. As shown in figure 2(a), this works much better than using all triples of nodes, as done by Kappen & Wiegerinck (2001). Note that if TreeEP is given a similar 'star' structure, the results are the same as GBP. This is because the GBP clusters coincide with the off-tree loops. In general, if the off-tree loops are triangles, then GBP on those triangles will give identical results.

Figure 2(b) shows the cost as $n$ increases. TreeEP and TreeVB scale the best, with TreeEP being the fastest method on large graphs.

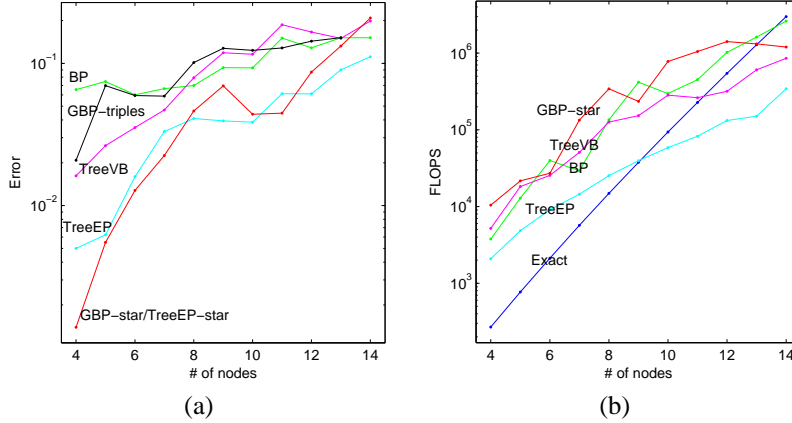

Figure 2: (a) Error in the estimated means for complete graphs with randomly chosen potentials. Each point is an average over 10 potentials. (b) Average FLOPS for the results in (a).

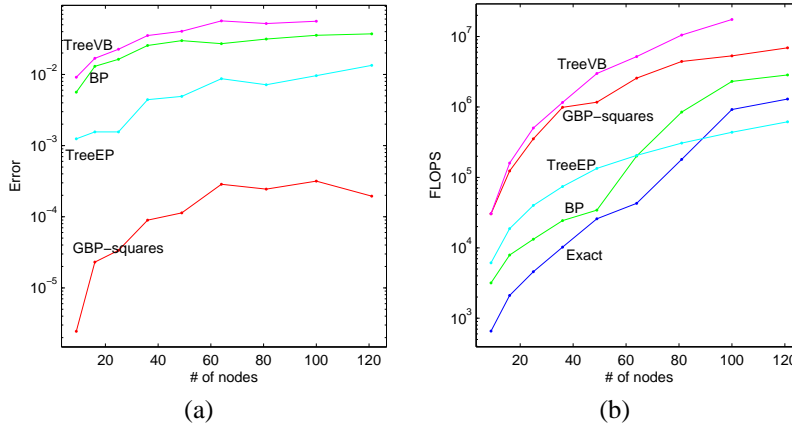

Figure 3: (a) Error in the estimated means for grid graphs with randomly chosen potentials. Each point is an average over 10 potentials. (b) Average FLOPS for the results in (a).

## 4.3   Grids

The next experiment tests the algorithms on square grids of varying size. The external fields $\theta_j$ were drawn as before, and the couplings $w_{jk}$ had standard deviation 1. The GBP clusters were overlapping squares, as in Yedidia et al. (2000).

Figure 3(a) shows the approximation error as $n$ increases, with results averaged over 10 trials as in the previous section. TreeVB performs consistently worse than BP, even though it is using the same tree structures as TreeEP. The plot also shows that these structures, being automatically chosen, are not as good as the hand-crafted clusters used by GBP. We have hand-crafted tree structures that perform just as well on grids, but for simplicity we do not include these results.

Figure 3(b) shows that TreeEP is the fastest on large grids, even faster than BP, because BP must use increasingly smaller stepsizes. GBP is more than a factor of ten slower.

# 5 Conclusions

Tree approximation allows a smooth tradeoff between cost and accuracy in approximate inference. It improves on BP for a modest increase in cost. In particular, when ordinary BP doesn't converge, TreeEP is an attractive alternative to damping or double-loop iteration. TreeEP performs better than the corresponding variational bounds, because it minimizes the inclusive KL-divergence. We found that TreeEP was equivalent to GBP in some cases, which deserves further study.

We hope that these results encourage more investigation into approximation structure for inference algorithms, such as finding the 'optimal' structure for a given problem. There are many other opportunities for special approximation structure to be exploited, especially in hybrid networks, where not only do the independence assumptions matter but also the distributional forms.

### Acknowledgments

We thank an anonymous reviewer for advice on comparisons to GBP.

# References

Chow, C. K., & Liu, C. N. (1968). Approximating discrete probability distributions with dependence trees. *IEEE Transactions on Information Theory*, *14*, 462–467.

Frey, B. J., Patrascu, R., Jaakkola, T., & Moran, J. (2000). Sequentially fitting inclusive trees for inference in noisy-OR networks. *NIPS 13*.

Ghahramani, Z., & Jordan, M. I. (1997). Factorial hidden Markov models. *Machine Learning*, *29*, 245–273.

Heskes, T., & Zoeter, O. (2002). Expectation propagation for approximate inference in dynamic Bayesian networks. *Proc UAI*.

Jensen, F. V., Lauritzen, S. L., & Olesen, K. G. (1990). Bayesian updating in causal probabilistic networks by local computations. *Computational Statistics Quarterly*, *5*, 269–282.

Kappen, H. J., & Wiegerinck, W. (2001). Novel iteration schemes for the cluster variation method. *NIPS 14*.

Minka, T. P. (2001a). Expectation propagation for approximate Bayesian inference. *UAI* (pp. 362–369).

Minka, T. P. (2001b). *A family of algorithms for approximate Bayesian inference*. Doctoral dissertation, Massachusetts Institute of Technology.

Murphy, K. (2001). The Bayes Net Toolbox for Matlab. *Computing Science and Statistics*, *33*.

Teh, Y. W., & Welling, M. (2001). The unified propagation and scaling algorithm. *NIPS 14*.

Wainwright, M. J., Jaakkola, T., & Willsky, A. S. (2001). Tree-based reparameterization for approximate estimation on loopy graphs. *NIPS 14*.

Wainwright, M. J., Jaakkola, T. S., & Willsky, A. S. (2002). A new class of upper bounds on the log partition function. *Proc UAI*.

Welling, M., & Teh, Y. W. (2001). Belief optimization for binary networks: A stable alternative to loopy belief propagation. *UAI*.

Wiegerinck, W. (2000). Variational approximations between mean field theory and the junction tree algorithm. *Proc UAI*.

Yedidia, J. S., Freeman, W. T., & Weiss, Y. (2000). Generalized belief propagation. *NIPS 13*.

Yedidia, J. S., Freeman, W. T., & Weiss, Y. (2002). Constructing free energy approximations and generalized belief propagation algorithms (Technical Report). MERL Research Lab.

Yuille, A. (In press, 2002). A double-loop algorithm to minimize the Bethe and Kikuchi free energies. *Neural Computation*.
